# Support Vector Regression Machines

**Harris Drucker\*    Chris J.C. Burges\*\*    Linda Kaufman\*\***
**Alex Smola\*\*    Vladimir Vapnik $^+$**

\*Bell Labs and Monmouth University
Department of Electronic Engineering
West Long Branch, NJ 07764
\*\*Bell Labs    $^+$AT&T Labs

## Abstract

A new regression technique based on Vapnik's concept of support vectors is introduced. We compare support vector regression (SVR) with a committee regression technique (bagging) based on regression trees and ridge regression done in feature space. On the basis of these experiments, it is expected that SVR will have advantages in high dimensionality space because SVR optimization does not depend on the dimensionality of the input space.

## 1. Introduction

In the following, lower case bold characters represent vectors and upper case bold characters represent matrices. Superscript "t" represents the transpose of a vector. y represents either a vector (in bold) or a single observance of the dependent variable in the presence of noise. $y^{(p)}$ indicates a predicted value due to the input vector $x^{(p)}$ not seen in the training set.

Suppose we have an unknown function $G(x)$ (the "truth") which is a function of a vector $x$ (termed *input space*). The vector $x^t = [x_1, x_2, ..., x_d]$ has d components where d is termed the dimensionality of the input space. $F(x, w)$ is a family of functions parameterized by $w$. $\hat{w}$ is that value of $w$ that minimizes a measure of error between $G(x)$ and $F(x, \hat{w})$. Our objective is to estimate $w$ with $\hat{w}$ by observing the $N$ training instances $v_j$, $j=1, \cdots, N$. We will develop two approximations for the truth $G(x)$. The first one is $F_1(x, \hat{w})$ which we term a feature space representation. One (of many) such feature vectors is:

$$z^t = [x_1^2, \cdots, x_d^2, x_1 x_2, \cdots, x_i x_j, \cdots, x_{d-1} x_d, x_1, \cdots, x_d, 1]$$

which is a quadratic function of the input space components. Using the feature space representation, then $F_1(x, \hat{w}) = z^t \hat{w}$, that is, $F_1(x, \hat{w})$ is linear in *feature space* although

it is quadratic in input space. In general, for a p'th order polynomial and d'th dimensional input space, the feature dimensionality $f$ of $\hat{w}$ is

$$f = \sum_{i=d-1}^{p+d-1} C_{d-1}^{i}$$

where $C_k^n = \dfrac{n!}{k!(n-k)!}$.

The second representation is a support vector regression (SVR) representation that was developed by Vladimir Vapnik (1995):

$$F_2(x,\hat{w})=\sum_{i=1}^{N}(\alpha_i^{*}-\alpha_i)(v_i^t x+1)^p + b$$

$F_2$ is an expansion explicitly using the training examples. The rationale for calling it a support vector representation will be clear later as will the necessity for having both an $\alpha^{*}$ and an $\alpha$ rather than just one multiplicative constant. In this case we must choose the $2N + 1$ values of $\alpha_i$ $\alpha_i^{*}$ and b. If we expand the term raised to the p'th power, we find $f$ coefficients that multiply the various powers and cross product terms of the components of $x$. So, in this sense $F_1$ looks very similar to $F_2$ in that they have the same number of terms. However $F_1$ has $f$ free coefficients while $F_2$ has $2N+1$ coefficients that must be determined from the N training vectors.

We let $\alpha$ represent the 2N values of $\alpha_i$ and $\alpha_i^{*}$. The optimum values for the components of $\hat{w}$ or $\alpha$ depend on our definition of the loss function and the objective function. Here the primal objective function is:

$$U\sum_{j=1}^{N}L[y_j-F(v_j,\hat{w})]+||\hat{w}||^2$$

where $L$ is a general loss function (to be defined later) and $F$ could be $F_1$ or $F_2$, $y_j$ is the observation of $G(x)$ in the presence of noise, and the last term is a regularizer. The regularization constant is $U$ which in typical developments multiplies the regularizer but is placed in front of the first term for reasons discussed later.

If the loss function is quadratic, i.e., we $L[\cdot]=[\cdot]^2$, and we let $F=F_1$, i.e., the feature space representation, the objective function may be minimized by using linear algebra techniques since the feature space representation is linear in that space. This is termed ridge regression (Miller, 1990). In particular let $V$ be a matrix whose i'th row is the i'th training vector represented in feature space (including the constant term "1" which represents a bias). $V$ is a matrix where the number of rows is the number of examples (N) and the number of columns is the dimensionality of feature space $f$. Let $E$ be the $f{\times}f$ diagonal matrix whose elements are $1/U$. $y$ is the Nx1 column vector of observations of the dependent variable. We then solve the following matrix formulation for $\hat{w}$ using a linear technique (Strang, 1986) with a linear algebra package (e.g., MATLAB):

$$V^t y = [V^t V + E]\,\hat{w}$$

The rationale for the regularization term is to trade off mean square error (the first term) in the objective function against the size of the $\hat{w}$ vector. If U is large, then essentially we are minimizing the mean square error on the training set which may give poor generalization to a test set. We find a good value of U by varying U to find the best performance on a validation set and then applying that U to the test set.

Let us now define a different type of loss function termed an $\varepsilon$-insensitive loss (Vapnik, 1995):

$$L = \begin{cases} 0 & \text{if } |y_i - F_2(x_i, \hat{w})| < \varepsilon \\ |y_i - F_2(x_i, \hat{w})| - \varepsilon & \text{otherwise} \end{cases}$$

This defines an $\varepsilon$ tube (Figure 1) so that if the predicted value is within the tube the loss is zero, while if the predicted point is outside the tube, the loss is the magnitude of the difference between the predicted value and the radius $\varepsilon$ of the tube.

Specifically, we minimize:

$$U\left(\sum_{i=1}^{N} \xi_i^* + \sum_{i=1}^{N} \xi_i\right) + \frac{1}{2}(w^t w)$$

where $\xi_i$ or $\xi_i^*$ is zero if the sample point is inside the tube. If the observed point is "above" the tube, $\xi_i$ is the positive difference between the observed value and $\varepsilon$ and $\alpha_i$ will be nonzero. Similary, $\xi_i^*$ will be nonzero if the observed point is below the tube and in this case $\alpha_i^*$ will be nonzero. Since an observed point can not be simultaneously on both sides of the tube, either $\alpha_i$ or $\alpha_i^*$ will be nonzero, unless the point is within the tube, in which case, both constants will be zero. If U is large, more emphasis is placed on the error while if U is small, more emphasis is placed on the norm of the weights leading to (hopefully) a better generalization. The constraints are: (for all i, i=1,N)

$$y_i - (w^t v_i) - b \leq \varepsilon + \xi_i$$
$$(w^t v_i) + b - y_i \leq \varepsilon + \xi_i^*$$
$$\xi_i^* \geq 0$$
$$\xi_i \geq 0$$

The corresponding Lagrangian is:

$$L = \frac{1}{2}(w^t w) + U\left(\sum_{i=1}^{N} \xi_i^* + \sum_{i=1}^{N} \xi_i\right) - \sum_{i=1}^{N} \alpha_i^* [y_i - (w^t v_i) - b + \varepsilon + \xi_i^*]$$
$$- \sum_{i=1}^{N} \alpha_i [(w^t v_i) + b - y_i + \varepsilon + \xi_i] - \sum_{i=1}^{N} (\gamma_i^* \xi_i^* + \gamma_i \xi_i)$$

where the $\gamma_i$ and $\alpha_i$ are Lagrange multipliers.

We find a saddle point of $L$ (Vapnik, 1995) by differentiating with respect to $w_i$, b, and $\xi$ which results in the equivalent maximization of the (dual space) objective function:

$$W(\alpha, \alpha^*) = -\varepsilon \sum_{i=1}^{N} (\alpha_i^* + \alpha_i) + \sum_{i=1}^{N} y_i (\alpha_i^* - \alpha_i) - \frac{1}{2} \sum_{i,j=1}^{N} (\alpha_i^* - \alpha_i)(\alpha_j^* - \alpha_j)(v_i^t v_j + 1)^p$$

with the constraints:

$$0 \leq \alpha_i \leq U \quad 0 \leq \alpha_i^* \leq U \quad i = 1, ..., N$$
$$\sum_{i=1}^{N} \alpha_i^* = \sum_{i=1}^{N} \alpha_i$$

We must find N Largrange multiplier pairs $(\alpha_i, \alpha_i^*)$. We can also prove that the product of $\alpha_i$ and $\alpha_i^*$ is zero which means that at least one of these two terms is zero. A $v_i$ corresponding to a non-zero $\alpha_i$ or $\alpha_i^*$ is termed a support vector. There can be at most N support vectors. Suppose now, we have a new vector $x^{(p)}$, then the corresponding

prediction of $y^{(p)}$ is:

$$y^{(p)} = \sum_{i=1}^{N}(\alpha_i^* - \alpha_i)(v_i^t x^{(p)} + 1)^P + b$$

Maximizing W is a quadratic programming problem but the above expression for W is not in standard form for use in quadratic programming packages (which usually does minimization). If we let

$$\beta_i = \alpha_i^* \qquad \beta_{i+N} = \alpha_i \quad i=1,...,N$$

then we minimize:

$$f(\beta) = \frac{1}{2}\beta^t Q\beta + c^t\beta$$

subject to the constraints

$$\sum_{i=1}^{N}\beta_i = \sum_{N+1}^{2N}\beta_i \quad \text{and} \quad 0 \le \beta_i \le U \quad i=1, \cdots, 2N$$

where

$$c^t = [\varepsilon - y_1, \varepsilon - y_2, \cdots, \varepsilon - y_N, \varepsilon + y_1, \varepsilon + y_2, \cdots, \varepsilon + y_N]$$

$$Q = \begin{bmatrix} D & -D \\ -D & D \end{bmatrix}$$

$$d_{ij} = (v_i^t v_j + 1)^P \quad i,j = 1, \cdots, N$$

We use an active set method (Bunch and Kaufman, 1980) to solve this quadratic programming problem.

## 2. Nonlinear Experiments

We tried three artificial functions from (Friedman, 1991) and a problem (Boston Housing) from the UCI database. Because the first three problems are artificial, we know both the observed values and the truths. Boston Housing has 506 cases with the dependent variable being the median price of housing in the Boston area. There are twelve continuous predictor variables. This data was obtaining from the UCI database (anonymous ftp at ftp.ics.uci.edu in directory /pub/machine-learning-databases) In this case, we have no "truth", only the observations.

In addition to the input space representation and the SVR representation, we also tried bagging. Bagging is a technique that combines regressors, in this case regression trees (Breiman, 1994). We used this technique because we had a local version available. In the case of regression trees, the validation set was used to prune the trees.

Suppose we have test points with input vectors $x_i^{(p)}$ $i=1,M$ and make a prediction $y_i^{(p)}$ using any procedure discussed here. Suppose $y_i$ is the actually observed value, which is the truth $G(x)$ plus noise. We define the prediction error (PE) and the modeling error (ME):

$$ME = \frac{1}{M}\sum_{i=1}^{M}(y_i^{(p)} - G(x_i))^2$$

$$PE = \frac{1}{M}\sum_{i=1}^{M}(y_i^{(p)} - y_i)^2$$

For the three Friedman functions we calculated both the prediction error and modeling

error. For Boston Housing, since the "truth" was not known, we calculated the prediction error only. For the three Friedman functions, we generated (for each experiment) 200 training set examples and 40 validation set examples. The validation set examples were used to find the optimum regularization constant in the feature space representation. The following procedure was followed. Train on the 200 members of the training set with a choice of regularization constant and obtain the prediction error on the validation set. Now repeat with a different regularization constant until a minimum of prediction error occurs on the validation set. Now, use that regularizer constant that minimizes the validation set prediction error and test on a 1000 example test set. This experiment was repeated for 100 different training sets of size 200 and validation sets of size 40 but one test set of size 1000. Different size polynomials were tried (maximum power 3). Second order polynomials fared best. For Friedman function #1, the dimensionality of feature space is 66 while for the last two problems, the dimensionality of feature space was 15 (for $d=2$). Thus the size of the feature space is smaller than that of the number of examples and we would expect that a feature space representation should do well.

A similar procedure was followed for the SVR representation except the regularizer constant U, $\varepsilon$ and power p were varied to find the minimum validation prediction error. In the majority of cases p=2 was the optimum choice of power.

For the Boston Housing data, we picked randomly from the 506 cases using a training set of size 401, a validation set of size 80 and a test set of size 25. This was repeated 100 times. The optimum power as picked by the validations set varied between p=4 and p=5.

### 3. Results of experiments

The first experiments we tried were bagging regression trees versus support regression (Table I).

Table I. Modeling error and prediction error
on the three Friedman problems (100 trials).

|  | bagging ME | SVR ME | bagging PE | SVR PE | # trials better |
|---|---|---|---|---|---|
| **#1** | 2.26 | .67 | 3.36 | 1.75 | 100 |
| **#2** | 10,185 | 4,944 | 66,077 | 60,424 | 92 |
| **#3** | .0302 | .0261 | .0677 | .0692 | 46 |

Rather than report the standard error, we did a comparison for each training set. That is, for the first experiment we tried both SVR and bagging on the same training, validation, and test set. If SVR had a better modeling error on the test set, it counted as a win. Thus for Friedman #1, SVR was always better than bagging on the 100 trials. There is no clear winner for Friedman function #3.

Subsequent to our comparison of bagging to SVR, we attempted working directly in feature space. That is, we used $F_1$ as our approximating function with square loss and a second degree polynomial. The results of this ridge regression (Table II) are better than SVR. In retrospect, this is not surprising since the dimensionality of feature space is small ($f=66$ for Friedman #1 and $f=15$ for the two remaining functions) in relation to the number of training examples (200). This was due to the fact that the best approximating polynomial is second order. The other advantages of the feature space representation in

this particular case are that both PE and ME are mean squared error and the loss function is mean squared error also.

Table II. Modeling error for SVR and
feature space polynomial approximation.

|     | SVR   | feature space |
|-----|-------|---------------|
| #1  | .67   | .61           |
| #2  | 4,944 | 3051          |
| #3  | .0261 | .0176         |

We now ask the question whether U and $\varepsilon$ are important in SVR by comparing the results in Table I with the results obtaining by setting $\varepsilon$ to zero and U to 100,000 making the regularizer insignificant (Table III). On Friedman #2 (and less so on Friedman #3), the proper choice of $\varepsilon$ and U are important.

Table III. Comparing the results above with those obtained by setting
$\varepsilon$ to zero and U to 100,000 (labeled suboptimum).

|     | optimum ME | suboptimum ME |
|-----|------------|---------------|
| #1  | .67        | .70           |
| #2  | 4,944      | 34,506        |
| #3  | .0261      | .0395         |

For the case of Boston Housing, the prediction error using bagging was 12.4 while for SVR we obtained 7.2 and SVR was better than bagging on 71 out of 100 trials. The optimum power seems to be about five. We never were able to get the feature representation to work well because the number of coefficients to be determined (6885) was much larger than the number of training examples (401).

## 4 Conclusions

Support vector regression was compared to bagging and a feature space representation on four nonlinear problems. On three of these problems a feature space representation was best, bagging was worst, and SVR came in second. On the fourth problem, Boston Housing, SVR was best and we were unable to construct a feature space representation because of the high dimensionality required of the feature space. On linear problems, forward subset selection seems to be the method of choice for the two linear problems we tried at varying signal to noise ratios.

In retrospect, the problems we decided to test on were too simple. SVR probably has greatest use when the dimensionality of the input space and the order of the approximation creates a dimensionality of a feature space representation much larger than that of the number of examples. This was not the case for the problems we considered. We thus need real life examples that fulfill these requirements.

## 5. Acknowledgements

This project was supported by ARPA contract number N00014-94-C-1086.

## 6. References

Leo Breiman, "Bagging Predictors", Technical Report 421, September 1994, Department of Statistics, University of California Berkeley, CA Also at anonymous ftp site: ftp.stat.berkeley.edu/pub/tech-reports/421.ps.Z.

Jame R. Bunch and Linda C. Kaufman, " A Computational Method of the Indefinite Quadratic Programming Problem", *Linear Algebra and Its Applications*, Elsevier-North Holland, 1980.

Jerry Friedman, "Multivariate Adaptive Regression Splines", *Annal of Statistics*, vol 19, No. 1, pp. 1-141

Alan J. Miller, *Subset Selection in Regression*, Chapman and Hall, 1990.

Gilbert Strang, *Introduction to Applied Mathematics*, Wellesley Cambridge Press, 1986.

Vladimir N. Vapnik, *The Nature of Statistical Learning Theory*, Springer, 1995.

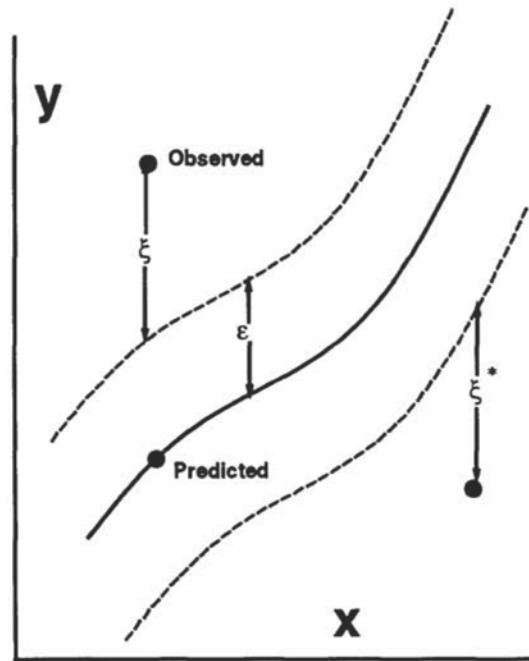

Figure 1: The parameters for the support vector regression.